# Distributed Recursive Structure Processing

**Géraldine Legendre**
Department of
Linguistics

**Yoshiro Miyata**
Optoelectronic
Computing Systems Center
University of Colorado
Boulder, CO 80309-0430*

**Paul Smolensky**
Department of
Computer Science

## Abstract

Harmonic grammar (Legendre, et al., 1990) is a connectionist theory of linguistic well-formedness based on the assumption that the well-formedness of a sentence can be measured by the harmony (negative energy) of the corresponding connectionist state. Assuming a lower-level connectionist network that obeys a few general connectionist principles but is otherwise unspecified, we construct a higher-level network with an equivalent harmony function that captures the most linguistically relevant global aspects of the lower level network. In this paper, we extend the tensor product representation (Smolensky 1990) to fully recursive representations of recursively structured objects like sentences in the lower-level network. We show theoretically and with an example the power of the new technique for parallel distributed structure processing.

## 1 Introduction

A new technique is presented for representing recursive structures in connectionist networks. It has been developed in the context of the framework of Harmonic Grammar (Legendre et al. 1990a, 1990b), a formalism for theories of linguistic well-formedness which involves two basic levels: At the lower level, elements of the problem domain are represented as distributed patterns of activity in a network; At the higher level, the elements in the domain are represented locally and connection weights are interpreted as soft rules involving these elements. There are two aspects that are central to the framework.

First, the connectionist well-formedness measure *harmony* (or negative "energy"), which we use to model linguistic well-formedness, has the properties that it is preserved between the lower and the higher levels and that it is maximized in the network processing. Our previous work developed techniques for deriving harmonies at the higher level from linguistic data, which allowed us to make contact with existing higher-level analyses of a given linguistic phenomenon.

This paper concentrates on the second aspect of the framework: how particular linguistic structures such as sentences can be efficiently represented and processed at the lower level. The next section describes a new method for representing tree structures in a network which is an extension of the tensor product representation proposed in (Smolensky 1990) that allows recursive tree structures to be represented and various tree operations to be performed in parallel.

## 2   Recursive tensor product representations

A *tensor product representation* of a set of structures $S$ assigns to each $s \in S$ a vector built up by superposing role-sensitive representations of its constituents. A *role decomposition* of $S$ specifies the constituent structure of $s$ by assigning to it an unordered set of *filler-role bindings*. For example, if $S$ is the set of strings from the alphabet $\{a, b, c\}$, and $s = cba$, then we might choose a role decomposition in which the roles are absolute positions in the string ($r_1 = $ first, $r_2 = $ second, ...) and the constituents are the filler/role bindings $\{b/r_2, a/r_3, c/r_1\}$. [1]

In a tensor product representation a constituent – i.e., a filler/role binding – is represented by the tensor (or generalized outer) product of vectors representing the filler and role in isolation: $f/r$ is represented by the vector $\mathbf{v} = \mathbf{f} \otimes \mathbf{r}$, which is in fact a second-rank tensor whose elements are conveniently labelled by two subscripts and defined simply by $\mathbf{v}_{\varphi\rho} = \mathbf{f}_\varphi \mathbf{r}_\rho$.

Where do the filler and role vectors $\mathbf{f}$ and $\mathbf{r}$ come from? In the most straightforward case, each filler is a member of a simple set $F$ (e.g. an alphabet) and each role is a member of a simple set $R$ and the designer of the representation simply specifies vectors representing all the elements of $F$ and $R$. In more complex cases, one or both of the sets $F$ and $R$ might be sets of structures which in turn can be viewed as having constituents, and which in turn can be represented using a tensor product representation. This recursive construction of the tensor product representations leads to tensor products of three or more vectors, creating tensors of rank three and higher, with elements conveniently labelled by three or more subscripts.

The recursive structure of trees leads naturally to such a recursive construction of a tensor product representation. (The following analysis builds on Section 3.7.2 of (Smolensky 1990).) We consider binary trees (in which every node has at most two children) since the techniques developed below generalize immediately to trees with higher branching factor, and since the power of binary trees is well attested, e.g., by the success of Lisp, whose basic datastructure is the binary tree. Adopting the conventions and notations of Lisp, we assume for simplicity that the terminal nodes

of the tree (those with no children), and only the terminal nodes, are labelled by symbols or atoms. The set of structures $S$ we want to represent is the union of a set of atoms and the set of binary trees with terminal nodes labelled by these atoms.

One way to view a binary tree, by analogy with how we viewed strings above, is as having a large number of positions with various locations relative to the root: we adopt *positional roles* $r_x$ labelled by binary strings (or bit vectors) such as $x = 0110$ which is the position in a tree accessed by "caddar = car(cdr(cdr(car)))", that is, the left child (0; car) of the right child (1; cdr) of the right child of the left child of the root of the tree. Using this role decomposition, each constituent of a tree is an atom (the filler) bound to some role $r_x$ specifying its location; so if a tree $s$ has a set of atoms $\{f_i\}$ at respective locations $\{x_i\}$, then the vector representing $s$ is $s = \sum_i f_i \otimes r_{x_i}$.

A more recursive view of a binary tree sees it as having only *two* constituents: the atoms or subtrees which are the left and right children of the root. In this *fully recursive role decomposition*, fillers may either be atoms or trees: the set of possible fillers $F$ is the same as the original set of structures $S$.

The fully recursive role decomposition can be incorporated into the tensor product framework by making the vector spaces and operations a little more complex than in (Smolensky 1990). The goal is a representation obeying, $\forall s, p, q \in S$:

$$s = \text{cons}(p, q) \Rightarrow s = p \otimes r_0 + q \otimes r_1 \qquad (1)$$

Here, $s = \text{cons}(p, q)$ is the tree with left subtree $p$ and right subtree $q$, while $s, p$ and $q$ are the vectors representing $s, p$ and $q$. The only two roles in this recursive decomposition are $r_0$, $r_1$: the left and right children of root. These roles are represented by two vectors $r_0$ and $r_1$.

A fully recursive representation obeying Equation 1 can actually be constructed from the positional representation, by assuming that the (many) positional role vectors are constructed recursively from the (two) fully recursive role vectors according to:

$$r_{x0} = r_x \otimes r_0 \qquad r_{x1} = r_x \otimes r_1.$$

For example, $r_{0110} = r_0 \otimes r_1 \otimes r_1 \otimes r_0$. [2] Thus the vectors representing positions at depth $d$ in the tree are tensors of rank $d$ (taking the root to be depth 0). As an example, the tree $s = \text{cons}(A, \text{cons}(B, C)) = \text{cons}(p, q)$, where $p = A$ and $q = \text{cons}(B, C)$, is represented by

$$
\begin{aligned}
s &= A \otimes r_0 + B \otimes r_{01} + C \otimes r_{11} = A \otimes r_0 + B \otimes r_0 \otimes r_1 + C \otimes r_1 \otimes r_1 \\
&= A \otimes r_0 + (B \otimes r_0 + C \otimes r_1) \otimes r_1 = p \otimes r_0 + q \otimes r_1,
\end{aligned}
$$

in accordance with Equation 1.

The complication in the vector spaces needed to accomplish this recursive analysis is one that allows us to add together the tensors of different ranks representing different depths in the tree. All we need do is take the direct sum of the spaces of tensors of different rank; in effect, concatenating into a long vector all the elements

of the tensors. For example, in $s = \text{cons}(A, \text{cons}(B, C))$, depth 0 is 0, since $s$ isn't an atom; depth 1 contains $A$, represented by the tensor $S^{(1)}_{\varphi\rho_1} = A_\varphi r0_{\rho_1}$, and depth 2 contains B and C, represented by $S^{(2)}_{\varphi\rho_1\rho_2} = B_\varphi r0_{\rho_1} r1_{\rho_2} + C_\varphi r1_{\rho_1} r1_{\rho_2}$. The tree as a whole is then represented by the sequence $s = \{S^{(0)}_\varphi, S^{(1)}_{\varphi\rho_1}, S^{(2)}_{\varphi\rho_1\rho_2}, \ldots\}$ where the tensor for depth 0, $S^{(0)}_\varphi$, and the tensors for depths $d > 2$, $S^{(d)}_{\varphi\rho_1\cdots\rho_d}$, are all zero.

We let $V$ denote the vector space of such sequences of tensors of rank 0, rank 1, ... , up to some maximum depth $D$ which may be infinite. Two elements of $V$ are added (or "superimposed") simply by adding together the tensors of corresponding rank. This is our vector space for representing trees. [3]

The vector operation **cons** for building the representation of a tree from that of its two subtrees is given by Equation 1. As an operation on $V$ this can be written:

$$\text{cons} : (\{P^{(0)}_\varphi, P^{(1)}_{\varphi\rho_1}, P^{(2)}_{\varphi\rho_1\rho_2}, \ldots\}, \{Q^{(0)}_\varphi, Q^{(1)}_{\varphi\rho_1}, Q^{(2)}_{\varphi\rho_1\rho_2}, \ldots\}) \mapsto$$

$$\{0, P^{(0)}_\varphi r0_{\rho_1}, P^{(1)}_{\varphi\rho_1} r0_{\rho_2}, \ldots\} + \{0, Q^{(0)}_\varphi r1_{\rho_1}, Q^{(1)}_{\varphi\rho_1} r1_{\rho_2}, \ldots\}$$

(Here, $0$ denotes the zero vector in the space representing atoms.) In terms of matrices multiplying vectors in $V$, this can be written

$$\text{cons}(p, q) = W_{\text{cons}0}\, p + W_{\text{cons}1}\, q$$

(parallel to Equation 1) where the non-zero elements of the matrix $W_{\text{cons}0}$ are

$$W_{\text{cons}0\,\varphi\rho_1\rho_2\cdots\rho_d\rho_{d+1},\varphi\rho_1\rho_2\cdots\rho_d} = r0_{\rho_{d+1}}$$

and $W_{\text{cons}1}$ is gotten by replacing $r_0$ with $r_1$.

Taking the car or cdr of a tree – extracting the left or right child – in the recursive decomposition is equivalent to *unbinding* either $r_0$ or $r_1$. As shown in (Smolensky 1990, Section 3.1), if the role vectors are linearly independent, this unbinding can be performed accurately, via a linear operation, specifically, a generalized inner product (tensor contraction) of the vector representing the tree with an unbinding vector $u_0$ or $u_1$. In general, the unbinding vectors are the dual basis to the role vectors; equivalently, they are the vectors comprising the inverse matrix to the matrix of all role vectors. If the role vectors are orthonormal (as in the simulation discussed below), the unbinding vectors are the same as the role vectors. The car operation can be written explicitly as an operation on $V$:

$$\text{car} : \{S^{(0)}_\varphi, S^{(1)}_{\varphi\rho}, S^{(2)}_{\varphi\rho_1\rho_1}, \ldots\} \mapsto$$

$$\{\textstyle\sum_{\rho_1} S^{(1)}_{\varphi\rho_1} u0_{\rho_1}, \sum_{\rho_2} S^{(2)}_{\varphi\rho_1\rho_2} u0_{\rho2}, \sum_{\rho_3} S^{(3)}_{\varphi\rho_1\rho_2\rho_3} u0_{\rho_3}, \ldots\}$$

[3] In the connectionist implementation simulated below, there is one unit for each element of each tensor in the sequence. In the simulation we report, seven atoms are represented by (binary) vectors in a three-dimensional space, so $\varphi = 0, 1, 2$; $r_0$ and $r_1$ are vectors in a two-dimensional space, so $\rho = 0, 1$. The number of units representing the portion of $V$ for depth $d$ is thus $3 \cdot 2^d$ and the total number of units representing depths up to $D$ is $3(2^{D+1} - 1)$. In tensor product representations, exact representation of deeply embedded structure does not come cheap.

(Replacing $u_0$ by $u_1$ gives cdr.) The operation **car** can be realized as a matrix $\mathbf{W}_{\mathbf{car}}$ mapping $V$ to $V$ with non-zero elements:

$$\mathbf{W}_{\mathbf{car}\,\varphi\rho_1\rho_2\cdots\rho_d,\varphi\rho_1\rho_2\cdots\rho_d\rho_{d+1}} = \mathbf{u}_{0\,\rho_{d+1}}.$$

$\mathbf{W}_{\mathbf{cdr}}$ is the same matrix, with $\mathbf{u}_0$ replaced by $\mathbf{u}_1$. [4]

One of the main points of developing this connectionist representation of trees is to enable massively parallel processing. Whereas in the traditional sequential implementation of Lisp, symbol processing consists of a long sequence of car, cdr, and cons operations, here we can compose together the corresponding sequence of $\mathbf{W}_{\mathbf{car}}$, $\mathbf{W}_{\mathbf{cdr}}$, $\mathbf{W}_{\mathbf{cons0}}$ and $\mathbf{W}_{\mathbf{cons1}}$ operations into a single matrix operation. Adding some minimal nonlinearity allows us to compose more complex operations incorporating the equivalent of conditional branching. We now illustrate this with a simple linguistically motivated example.

## 3   An example

The symbol manipulation problem we consider is that of transforming a tree representation of a syntactic parse of an English sentence into a tree representation of a predicate-calculus expression for the meaning of the sentence. We considered two possible syntactic structures: simple active sentences of the form  and passive sentences of the form . Each was to be transformed into a tree representing V(A,P), namely . Here, the agent △ⒶA and patient △⒫P of the verb V are both arbitrarily complex noun phrase trees. (Actually, the network could handle arbitrarily complex V's as well.) Aux is a marker for passive (eg. *is* in *is feared.*)

The network was presented with an input tree of either type, represented as an activation vector using the fully recursive tensor product representation developed in the preceding section. The seven non-zero binary vectors of length three coded seven atoms; the role vectors used were technique described above. The desired output was the same tensorial representation of the tree representing V(A, B). The filler vectors for the verb and for the constituent words of the two noun phrases should be unbound from their roles in the input tree and then bound to the appropriate roles in the output tree.

Such transformation was performed, for an active sentence, by the operation cons(cadr(s), cons(car(s), cddr(s))) on the input tree s, and for a passive sentence, by cons(cdadr(s), cons(cdddr(s), car(s))). These operations were implemented in the network as two weight matrices, $\mathbf{W}_a$ and $\mathbf{W}_p$, [5] connecting the input units to the output units as shown in Figure 1. In addition, the network had a circuit for

Output = cons(V,cons(C,cons(A,B)))

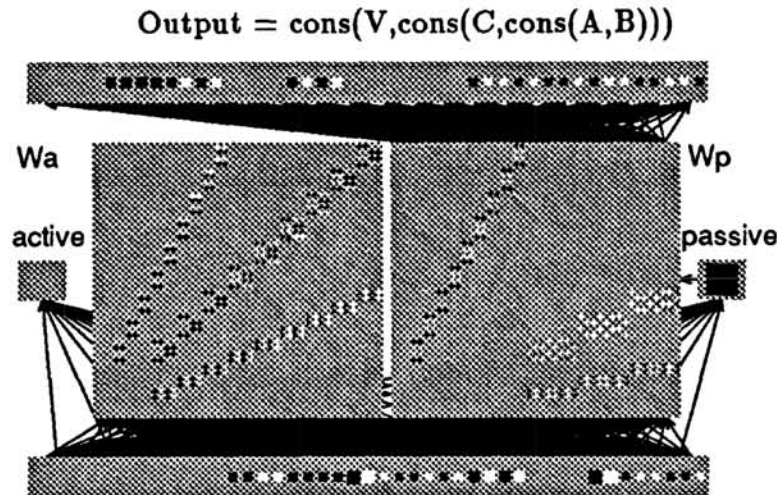

Input = cons(cons(A,B),cons(cons(Aux,V),cons(by,C)))

Figure 1: Recursive tensor product network processing a passive sentence

determining whether the input sentence was active or passive. In this example, it simply computed, by a weight matrix, the caddr of the input tree (where a passive sentence should have an Aux), and if it was the marker Aux, gated (with sigma-pi connections) $\mathbf{W_p}$, and otherwise gated $\mathbf{W_a}$.

Given this setting, the network was able to process arbitrary input sentences of either type, up to a certain depth (4 in this example) limited by the size of the network, properly and generated correct case role assignments. Figure 1 shows the network processing a passive sentence ((A.B).((Aux.V).(by.C))) as in *All connectionists are feared by Minsky* and generating (V.(C.(A.B))) as output.

## 4    Discussion

The formalism developed here for the recursive representation of trees generates quite different representations depending on the choice of the two fundamental role vectors $r_0$ and $r_1$ and the vectors for representing the atoms. At one extreme is the trivial fully local representation in which one connectionist unit is dedicated to each possible atom in each possible position: this is the special case in which $r_0$ and $r_1$ are chosen to be the canonical basis vectors (1 0) and (0 1), and the vectors representing the $n$ atoms are also chosen to be the canonical basis vectors of $n$-space. The example of the previous section illustrated the case of (a) linearly dependent vectors for atoms and (b) orthonormal vectors for the roles that were "distributed" in that both elements of both vectors were non-zero. Property (a) permits the representation of many more than $n$ atoms with $n$-dimensional vectors, and could be used to enrich the usual notions of symbolic computation by letting "similar atoms" be represented by vectors that are closer to each other than are "dissimilar atoms." Property (b) contributes no savings in units of the purely local case, amounting to a literal rotation in role space. But it does allow us

to demonstrate that fully distributed representations are as capable as fully local ones at supporting massively parallel structure processing. This point has been denied (often rather loudly) by advocates of local representations and by such critics as (Fodor & Pylyshyn 1988) and (Fodor & McLaughlin 1990) who have claimed that only connectionist implementations that preserve the concatenative structure of language-like representations of symbolic structures could be capable of true structure-sensitive processing.

The case illustrated in our example is distributed in the sense that all units corresponding to depth $d$ in the tree are involved in the representation of all the atoms at that depth. But different depths are kept separate in the formalism and in the network. We can go further by allowing the role vectors to be linearly dependent, sacrificing full accuracy and generality in structure processing for representation of greater depth in fewer units. This case is the subject of current research, but space limitations have prevented us from describing our preliminary results here.

Returning to Harmonic Grammar, the next question is, having developed a fully recursive tensor product representation for lower-level representation of embedded structures such as those ubiquitous in syntax, what are the implications for well-formedness as measured by the harmony function? A first approximation to the natural language case is captured by context free grammars, in which the well-formedness of a subtree is independent of its level of embedding. It turns out that such depth-independent well-formedness is captured by a simple equation governing the harmony function (or weight matrix). At the higher level where grammatical "rules" of Harmonic Grammar reside, this has the consequence that the numerical constant appearing in each soft constraint that constitutes a "rule" applies at all levels of embedding. This greatly constrains the parameters in the grammar.

## Footnotes

*The authors are listed in alphabetical order.

[1]The other major kind of role decomposition considered in (Smolensky 1990) is contextual roles; under one such decomposition, one constituent of cba is "b in the role 'preceded by c and followed by a'".

[2] By adopting this definition of $r_x$, we are essentially taking the recursive structure that is implicit in the subscripts $x$ labelling the positional role vectors, and mapping it into the structure of the vectors themselves.

[4]Note that in the case when the $\{\mathbf{r}_0,\mathbf{r}_1\}$ are orthonormal, and therefore $\mathbf{u}_0 = \mathbf{r}_0$, $\mathbf{W}_{\mathbf{car}} = \mathbf{W}_{\mathbf{cons0}}{}^{\mathbf{T}}$; similarly, $\mathbf{W}_{\mathbf{cdr}} = \mathbf{W}_{\mathbf{cons1}}{}^{\mathbf{T}}$.

[5]The two weight matrices were constructed from the four basic matrices as $\mathbf{W}_a = \mathbf{W}_{\mathbf{cons0}}\mathbf{W}_{\mathbf{car}}\mathbf{W}_{\mathbf{cdr}} + \mathbf{W}_{\mathbf{cons1}}(\mathbf{W}_{\mathbf{cons0}}\mathbf{W}_{\mathbf{car}} + \mathbf{W}_{\mathbf{cons1}}\mathbf{W}_{\mathbf{cdr}}\mathbf{W}_{\mathbf{cdr}})$ and $\mathbf{W}_p = \mathbf{W}_{\mathbf{cons0}}\mathbf{W}_{\mathbf{cdr}}\mathbf{W}_{\mathbf{car}}\mathbf{W}_{\mathbf{cdr}} + \mathbf{W}_{\mathbf{cons1}}(\mathbf{W}_{\mathbf{cons0}}\mathbf{W}_{\mathbf{cdr}}\mathbf{W}_{\mathbf{cdr}}\mathbf{W}_{\mathbf{cdr}} + \mathbf{W}_{\mathbf{cons1}}\mathbf{W}_{\mathbf{car}})$.

## References

[1] J. A. Fodor and B. P. McLaughlin. Connectionism and the problem of systematicity: Why smolensky's solution doesn't work. *Cognition*, 35:183–204, 1990.

[2] J. A. Fodor and Z. W. Pylyshyn. Connectionism and cognitive architecture: A critical analysis. *Cognition*, 28:3–71, 1988.

[3] G. Legendre, Y. Miyata, and P. Smolensky. Harmonic grammar – a formal multi-level connectionist theory of linguistic well-formedness: Theoretical foundations. In *the Proceedings of the twelveth meeting of the Cognitive Science Society*, 1990a.

[4] G. Legendre, Y. Miyata, and P. Smolensky. Harmonic grammar – a formal multi-level connectionist theory of linguistic well-formedness: An application. In *the Proceedings of the twelveth meeting of the Cognitive Science Society*, 1990b.

[5] P. Smolensky. Tensor product variable binding and the representation of symbolic structures in connectionist networks. *Artificial Intelligence*, 46:159–216, 1990.
